# Geometry of Early Stopping in Linear Networks

**Robert Dodier** *
Dept. of Computer Science
University of Colorado
Boulder, CO 80309

## Abstract

A theory of early stopping as applied to linear models is presented. The backpropagation learning algorithm is modeled as gradient descent in continuous time. Given a training set and a validation set, all weight vectors found by early stopping must lie on a certain quadric surface, usually an ellipsoid. Given a training set and a candidate early stopping weight vector, all validation sets have least-squares weights lying on a certain plane. This latter fact can be exploited to estimate the probability of stopping at any given point along the trajectory from the initial weight vector to the least-squares weights derived from the training set, and to estimate the probability that training goes on indefinitely. The prospects for extending this theory to nonlinear models are discussed.

## 1 INTRODUCTION

'Early stopping' is the following training procedure:

> Split the available data into a training set and a "validation" set. Start with initial weights close to zero. Apply gradient descent (backpropagation) on the training data. If the error on the validation set increases over time, stop training.

This training method, as applied to neural networks, is of relatively recent origin. The earliest references include Morgan and Bourlard [4] and Weigend *et al.* [7].

Finnoff *et al.* [2] studied early stopping empirically. While the goal of a theory of early stopping is to analyze its application to nonlinear approximators such as sigmoidal networks, this paper will deal mainly with linear systems and only marginally with nonlinear systems. Baldi and Chauvin [1] and Wang *et al.* [6] have also analyzed linear systems.

The main result of this paper can be summarized as follows. It can be shown (see Sec. 5) that the most probable stopping point on a given trajectory (fixing the training set and initial weights) is the same no matter what the size of the validation set. That is, the most probable stopping point (considering all possible validation sets) for a finite validation set is the same as for an infinite validation set. (If the validation data is unlimited, then the validation error is the same as the true generalization error.) However, for finite validation sets there is a dispersion of stopping points around the best (most probable and least generalization error) stopping point, and this increases the expected generalization error. See Figure 1 for an illustration of these ideas.

## 2   MATHEMATICAL PRELIMINARIES

In what follows, backpropagation will be modeled as a process in continuous time. This corresponds to letting the learning rate approach zero. This continuum model simplifies the necessary algebra while preserving the important properties of early stopping. Let the inputs be denoted $\mathbf{X} = (x_{ij})$, so that $x_{ij}$ is the $j$'th component of the $i$'th observation; there are $p$ components of each of the $n$ observations. Likewise, let $\mathbf{y} = (y_i)$ be the (scalar) outputs observed when the inputs are $\mathbf{X}$. Our regression model will be a linear model, $y_i = \mathbf{w}'\mathbf{x}_i + \epsilon_i$, $i = 1, \ldots, n$. Here $\epsilon_i$ represents independent, identically distributed (i.i.d.) Gaussian noise, $\epsilon_i \sim N(0, \sigma^2)$. Let $E(\mathbf{w}) = \frac{1}{2}\|\mathbf{X}\mathbf{w} - \mathbf{y}\|^2$ be one–half the usual sum of squared errors.

The error gradient with respect to the weights is $\nabla E(\mathbf{w}) = \mathbf{w}'\mathbf{X}'\mathbf{X} - \mathbf{y}'\mathbf{X}$. The backprop algorithm is modeled as $\dot{\mathbf{w}} = -\nabla E(\mathbf{w})$. The least-squares solution, at which $\nabla E(\mathbf{w}) = 0$, is $\mathbf{w}_{LS} = (\mathbf{X}'\mathbf{X})^{-1}\mathbf{X}'\mathbf{y}$. Note the appearence here of the input correlation matrix, $\mathbf{X}'\mathbf{X} = (\sum_{k=1}^{n} x_{ki}x_{kj})$. The properties of this matrix determine, to a large extent, the properties of the least-squares solutions we find. It turns out that as the number of observations $n$ increases without bound, the matrix $\sigma^2(\mathbf{X}'\mathbf{X})^{-1}$ converges with probability one to the population covariance matrix of the weights. We will find that the correlation matrix plays an important role in the analysis of early stopping.

We can rewrite the error $E$ using a diagonalization of the correlation matrix $\mathbf{X}'\mathbf{X} = \mathbf{S}\Lambda\mathbf{S}'$. Omitting a few steps of algebra,

$$E(\mathbf{w}) = \tfrac{1}{2}\sum_{k=1}^{p} \lambda_k v_k^2 + \tfrac{1}{2}\mathbf{y}'(\mathbf{y} - \mathbf{X}\mathbf{w}_{LS}) \tag{1}$$

where $\mathbf{v} = \mathbf{S}'(\mathbf{w} - \mathbf{w}_{LS})$ and $\Lambda = \mathrm{diag}(\lambda_1, \ldots, \lambda_p)$. In this sum we see that the magnitude of the $k$'th term is proportional to the corresponding characteristic value, so moving $\mathbf{w}$ toward $\mathbf{w}_{LS}$ in the direction corresponding to the largest characteristic value yields the greatest reduction of error. Likewise, moving in the direction corresponding to the smallest characteristic value gives the least reduction of error.

So far, we have implicitly considered only one set of data; we have assumed all data is used for training. Now let us distinguish training data, $\mathbf{X}_t$ and $\mathbf{y}_t$, from validation data, $\mathbf{X}_v$ and $\mathbf{y}_v$; there are $n_t$ training and $n_v$ validation data. Now each set of data has its own least-squares weight vector, $\mathbf{w}_t$ and $\mathbf{w}_v$, and its own error gradient, $\nabla E_t(\mathbf{w})$ and $\nabla E_v(\mathbf{w})$. Also define $\mathbf{M}_t = \mathbf{X}_t'\mathbf{X}_t$ and $\mathbf{M}_v = \mathbf{X}_v'\mathbf{X}_v$ for convenience. The early stopping method can be analyzed in terms of the these pairs of matrices, gradients, and least-squares weight vectors.

## 3   THE MAGIC ELLIPSOID

Consider the early stopping criterion, $\frac{dE_v}{dt}(\mathbf{w}) = 0$. Applying the chain rule,

$$\frac{dE_v}{dt} = \frac{dE_v}{d\mathbf{w}} \cdot \frac{d\mathbf{w}}{dt} = \nabla E_v \cdot -\nabla E_t, \tag{2}$$

where the last equality follows from the definition of gradient descent. So the early stopping criterion is the same as saying

$$\nabla E_t \cdot \nabla E_v = 0, \tag{3}$$

that is, at an early stopping point, the training and validation error gradients are perpendicular, if they are not zero.

Consider now the set of all points in the weight space such that the training and validation error gradients are perpendicular. These are the points at which early stopping may stop. It turns out that this set of points has an easily described shape. The condition given by Eq. 3 is equivalent to

$$0 = \nabla E_t \cdot \nabla E_v = (\mathbf{w} - \mathbf{w}_t)'\mathbf{M}_t\mathbf{M}_v'(\mathbf{w} - \mathbf{w}_v). \tag{4}$$

Note that all correlation matrices are symmetric, so $\mathbf{M}_t\mathbf{M}_v' = \mathbf{M}_t\mathbf{M}_v$. We see that Eq. 4 gives a quadratic form. Let us put Eq. 4 into a standard form. Toward this end, let us define some useful terms. Let

$$\begin{aligned}
\mathbf{M} &= \mathbf{M}_t\mathbf{M}_v, & (5) \\
\bar{\mathbf{M}} &= \tfrac{1}{2}(\mathbf{M} + \mathbf{M}') = \tfrac{1}{2}(\mathbf{M}_t\mathbf{M}_v + \mathbf{M}_v\mathbf{M}_t), & (6) \\
\bar{\mathbf{w}} &= \tfrac{1}{2}(\mathbf{w}_t + \mathbf{w}_v), & (7) \\
\Delta\mathbf{w} &= \mathbf{w}_t - \mathbf{w}_v, & (8)
\end{aligned}$$

and

$$\widetilde{\mathbf{w}} = \bar{\mathbf{w}} - \tfrac{1}{4}\bar{\mathbf{M}}^{-1}(\mathbf{M} - \mathbf{M}')\Delta\mathbf{w}. \tag{9}$$

Now an important result can be stated. The proof is omitted.

**Proposition 1.** $\nabla E_t \cdot \nabla E_v = 0$ is equivalent to

$$(\mathbf{w} - \widetilde{\mathbf{w}})'\bar{\mathbf{M}}(\mathbf{w} - \widetilde{\mathbf{w}}) = \tfrac{1}{4}\Delta\mathbf{w}[\bar{\mathbf{M}} + \tfrac{1}{4}(\mathbf{M}' - \mathbf{M})\bar{\mathbf{M}}^{-1}(\mathbf{M} - \mathbf{M}')]\Delta\mathbf{w}. \quad \square \tag{10}$$

The matrix $\bar{\mathbf{M}}$ of the quadratic form given by Eq. 10 is "usually" positive definite. As the number of observations $n_t$ and $n_v$ of training and validation data increase without bound, $\bar{\mathbf{M}}$ converges to a positive definite matrix. In what follows it will

always be assumed that $\bar{\mathbf{M}}$ is indeed positive definite. Given this, the locus defined by $\nabla E_t \perp \nabla E_v$ is an ellipsoid. The centroid is $\widetilde{\mathbf{w}}$, the orientation is determined by the characteristic vectors of $\bar{\mathbf{M}}$, and the length of the $k$'th semiaxis is $\sqrt{c/\bar{\lambda}_k}$, where $c$ is the constant on the righthand side of Eq. 10 and $\bar{\lambda}_k$ is the $k$'th characteristic value of $\bar{\mathbf{M}}$.

## 4  THE MAGIC PLANE

Given the least-squares weight vector $\mathbf{w}_t$ derived from the training data and a candidate early stopping weight vector $\mathbf{w}_{es}$, any least-squares weight vector $\mathbf{w}_v$ from a validation set must lie on a certain plane, the 'magic plane.' The proof of this statement is omitted.

**Proposition 2.** The condition that $\mathbf{w}_t$, $\mathbf{w}_v$, and $\mathbf{w}_{es}$ all lie on the magic ellipsoid,

$$(\mathbf{w}_t - \widetilde{\mathbf{w}})'\bar{\mathbf{M}}(\mathbf{w}_t - \widetilde{\mathbf{w}}) = (\mathbf{w}_v - \widetilde{\mathbf{w}})'\bar{\mathbf{M}}(\mathbf{w}_v - \widetilde{\mathbf{w}}) = (\mathbf{w}_{es} - \widetilde{\mathbf{w}})'\bar{\mathbf{M}}(\mathbf{w}_{es} - \widetilde{\mathbf{w}}) = c, \quad (11)$$

implies

$$(\mathbf{w}_t - \mathbf{w}_{es})'\mathbf{M}\mathbf{w}_v = (\mathbf{w}_t - \mathbf{w}_{es})'\mathbf{M}\mathbf{w}_{es}. \quad \square \qquad (12)$$

This shows that $\mathbf{w}_v$ lies on a plane, the magic plane, with normal $\mathbf{M}'(\mathbf{w}_t - \mathbf{w}_{es})$. The reader will note a certain difficulty here, namely that $\mathbf{M} = \mathbf{M}_t\mathbf{M}_v$ depends on the particular validation set used, as does $\mathbf{w}_v$. However, we can make progress by considering only a fixed correlation matrix $\mathbf{M}_v$ and letting $\mathbf{w}_v$ vary. Let us suppose the inputs $(x_1, x_2, \ldots, x_p)$ are i.i.d. Gaussian random variables with mean zero and some covariance $\mathbf{\Sigma}$. (Here the inputs are random but they are observed exactly, so the error model $y = \mathbf{w}'\mathbf{x} + \epsilon$ still applies.) Then

$$\langle \mathbf{M}_v \rangle = \langle \mathbf{X}_v'\mathbf{X}_v \rangle = n_v\mathbf{\Sigma},$$

so in Eq. 12 let us replace $\mathbf{M}_v$ with its expected value $n_v\mathbf{\Sigma}$. That is, we can approximate Eq. 12 with

$$(\mathbf{w}_t - \mathbf{w}_{es})'\mathbf{M}_t\mathbf{\Sigma}\mathbf{w}_v = (\mathbf{w}_t - \mathbf{w}_{es})'\mathbf{M}_t\mathbf{\Sigma}\mathbf{w}_{es}. \qquad (13)$$

Now consider the probability that a particular point $\mathbf{w}(t)$ on the trajectory from $\mathbf{w}(0)$ to $\mathbf{w}_t$ is an early stopping point, that is, $\nabla E_t(\mathbf{w}(t)) \cdot \nabla E_v(\mathbf{w}(t)) = 0$. This is exactly the probability that Eq. 12 is satisfied, and approximately the probability that Eq. 13 is satisfied. This latter approximation is easy to calculate: it is the mass of an infinitesimally–thin slab cutting through the distribution of least-squares validation weight vectors. Given the usual additive noise model $y = \mathbf{w}'\mathbf{x} + \epsilon$ with $\epsilon$ being i.i.d. Gaussian distributed noise with mean zero and variance $\sigma^2$, the least-squares weights are approximately distributed as

$$\mathbf{w} - \mathbf{w}^* \sim N(0, \sigma^2(\mathbf{X}'\mathbf{X})^{-1}) \qquad (14)$$

when the number of data is large.

Consider now the plane $\Omega = \{\mathbf{w} : \mathbf{w}'\hat{\mathbf{n}} = k\}$. The probability mass on this plane as it cuts through a Gaussian distribution $N(\mu, \mathbf{C})$ is then

$$p_\Omega(k, \hat{\mathbf{n}}) = (2\pi\hat{\mathbf{n}}'\mathbf{C}\hat{\mathbf{n}})^{-1/2}\exp(-\frac{1}{2}\frac{(k - \mu'\hat{\mathbf{n}})^2}{\hat{\mathbf{n}}'\mathbf{C}\hat{\mathbf{n}}})\,ds \qquad (15)$$

where $ds$ denotes an infinitesimal arc length. (See, for example, Sec. VIII-9.3 of von Mises [3].)

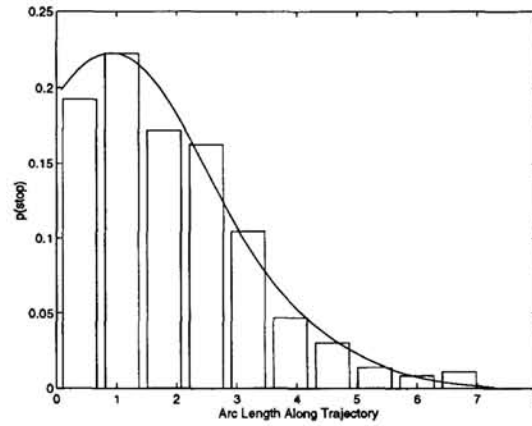

Figure 1: Histogram of early stopping points along a trajectory, with bins of equal arc length. An approximation to the probability of stopping (Eq. 16) is superimposed. Altogether 1000 validation sets were generated for a certain training set; of these, 288 gave "don't start" solutions, 701 gave early stopping solutions (which are binned here) somewhere on the trajectory, and 11 gave "don't stop" solutions.

## 5  PROBABILITY OF STOPPING AT A GIVEN POINT

Let us apply Eq. 15 to the problem at hand. Our normal is $\hat{\mathbf{n}} = n_v \mathbf{\Sigma M}_t(\mathbf{w}_t - \mathbf{w}_{es})$ and the offset is $k = \hat{\mathbf{n}}' \mathbf{w}_{es}$. A formal statement of the approximation of $p_\Omega$ can now be made.

**Proposition 3.** Assuming the validation correlation matrix $\mathbf{X}'_v \mathbf{X}_v$ equals the mean correlation matrix $n_v \mathbf{\Sigma}$, the probability of stopping at a point $\mathbf{w}_{es} = \mathbf{w}(t)$ on the trajectory from $\mathbf{w}(0)$ to $\mathbf{w}_t$ is approximately

$$p_\Omega(t) = p_\Omega(k(t), \hat{\mathbf{n}}(t)) = (2\pi \hat{\mathbf{n}}' \mathbf{C} \hat{\mathbf{n}})^{-1/2} \exp(-\frac{1}{2} \frac{(\hat{\mathbf{n}}'(\mathbf{w}_{es} - \mathbf{w}^*))^2}{\hat{\mathbf{n}}' \mathbf{C} \hat{\mathbf{n}}}), \qquad (16)$$

with

$$\hat{\mathbf{n}}' \mathbf{C} \hat{\mathbf{n}} = n_v \sigma^2 (\mathbf{w}_t - \mathbf{w}_{es})' \mathbf{M}_t \mathbf{\Sigma M}_t(\mathbf{w}_t - \mathbf{w}_{es}). \ \square \qquad (17)$$

How useful is this approximation? Simulations were carried out in which the initial weight vector $\mathbf{w}(0)$ and the training data ($n_t = 20$) were fixed, and many validation sets of size $n_v = 20$ were generated (without fixing $\mathbf{X}'_v \mathbf{X}_v$). The trajectory was divided into segments of equal length and histograms of the number of early stopping weights on each segment were constructed. A typical example is shown in Figure 1. It can be seen that the empirical histogram is well-approximated by Eq. 16.

If for some $\mathbf{w}(t)$ on the trajectory the magic plane cuts through the true weights $\mathbf{w}^*$, then $p_\Omega$ will have a peak at $t$. As the number of validation data $n_v$ increases, the variance of $\mathbf{w}_v$ decreases and the peak narrows, but the position $\mathbf{w}(t)$ of the peak does not move. As $n_v \to \infty$ the peak becomes a spike at $\mathbf{w}(t)$. That is, the peak of $p_\Omega$ for a finite validation set is the same as if we had access to the true generalization error. In this sense, early stopping does the right thing.

It has been observed that when early stopping is employed, the validation error may decrease forever and never rise – thus the 'early stopping' procedure yields the least-squares weights. How common is this phenomenon? Let us consider a fixed

training set and a fixed initial weight vector, so that the trajectory is fixed. Letting the validation set range over all possible realizations, let us denote by $P_\Omega(t) = P_\Omega(k(t), \hat{\mathbf{n}}(t))$ the probability that training stops at time $t$ or later. $1 - P_\Omega(0)$ is the probability that validation error rises immediately upon beginning training, and let us agree that $P_\Omega(\infty)$ denotes the probability that validation error never increases. This $P_\Omega(t)$ is approximately the mass that is "behind" the plane $\hat{\mathbf{n}}'\mathbf{w}_v = \hat{\mathbf{n}}'\mathbf{w}_{es}$, "behind" meaning the points $\mathbf{w}_v$ such that $(\mathbf{w}_v - \mathbf{w}_{es})'\hat{\mathbf{n}} < 0$. (The identification of $P_\Omega$ with the mass to one side of the plane is not exact because intersections of magic planes are ignored.) As Eq. 15 has the form of a Gaussian p.d.f., it is easy to show that

$$P_\Omega(k, \hat{\mathbf{n}}) = G\left(\frac{k - \hat{\mathbf{n}}'\mathbf{w}^*}{(\hat{\mathbf{n}}'\mathbf{C}\hat{\mathbf{n}})^{1/2}}\right) \tag{18}$$

where $G$ denotes the standard Gaussian c.d.f., $G(z) = (2\pi)^{-1/2} \int_{-\infty}^{z} \exp(-t^2/2)dt$. Recall that we take the normal $\hat{\mathbf{n}}$ of the magic plane through $\mathbf{w}_{es}$ as $\hat{\mathbf{n}} = \mathbf{\Sigma}\mathbf{M}_t(\mathbf{w}_t - \mathbf{w}_{es})$. For $t = 0$ there is no problem with Eq. 18 and an approximation for the "never-starting" probability is stated in the next proposition.

**Proposition 4.** The probability that validation error increases immediately upon beginning training ("never starting"), assuming the validation correlation matrix $\mathbf{X}_v'\mathbf{X}_v$ equals the mean correlation matrix $n_v\mathbf{\Sigma}$, is approximately

$$1 - P_\Omega(0) = 1 - G\left(\frac{\sqrt{n_v}}{\sigma} \frac{(\mathbf{w}^* - \mathbf{w}(0))'\mathbf{M}_t\mathbf{\Sigma}(\mathbf{w}_t - \mathbf{w}(0))}{[(\mathbf{w}_t - \mathbf{w}(0))'\mathbf{M}_t\mathbf{\Sigma}\mathbf{M}_t(\mathbf{w}_t - \mathbf{w}(0))]^{1/2}}\right). \quad \Box \tag{19}$$

With similar arguments we can develop an approximation to the "never-stopping" probability.

**Proposition 5.** The probability that training continues indefinitely ("never stopping"), assuming the validation correlation matrix $\mathbf{X}_v'\mathbf{X}_v$ equals the mean correlation matrix $n_v\mathbf{\Sigma}$, is approximately

$$P_\Omega(\infty) \;=\; G\left(\frac{\sqrt{n_v}}{\sigma} \frac{(\mathbf{w}^* - \mathbf{w}_t)'\mathbf{M}_t\mathbf{\Sigma}(\pm\mathbf{s}^*)}{\lambda^*[(\mathbf{s}^*)'\mathbf{\Sigma}\mathbf{s}^*]^{1/2}}\right). \tag{20}$$

In Eq. 20 pick $+\mathbf{s}^*$ if $(\mathbf{w}_t - \mathbf{w}(0))'\mathbf{s}^* > 0$, otherwise pick $-\mathbf{s}^*$. $\quad \Box$

Simulations are in good agreement with the estimates given by Propositions 4 and 5.

# 6   EXTENDING THE THEORY TO NONLINEAR SYSTEMS

It may be possible to extend the theory presented in this paper to nonlinear approximators. The elementary concepts carry over unchanged, although it will be more difficult to describe them algebraically. In a nonlinear early stopping problem, there will be a surface corresponding to the magic ellipsoid on which $\nabla E_t \perp \nabla E_v$, but this surface may be nonconvex or not simply connected. Likewise, corresponding to the magic plane there will be a surface on which least-squares validation weights must fall, but this surface need not be flat or unbounded.

It is customary in the world of statistics to apply results derived for linear systems to nonlinear systems by assuming the number of data is very large and various

regularity conditions hold. If the errors $\epsilon$ are additive, the least-squares weights again have a Gaussian distribution. As in the linear case, the Hessian of the total error appears as the inverse of the covariance of the least-squares weights. In this asymptotic (large data) regime, the standard results for linear regression carry over to nonlinear regression mostly unchanged. This suggests that the linear theory of early stopping will also apply to nonlinear regression models, such as sigmoidal networks, when there is much data.

However, it should be noted that the asymptotic regression theory is purely local – it describes only what happens in the neighborhood of the least-squares weights. As the outcome of early stopping depends upon the initial weights and the trajectory taken through the weight space, any local theory will not suffice to analyze early stopping. Nonlinear effects such as local minima and non-quadratic basins cannot be accounted for by a linear or asymptotically linear theory, and these may play important roles in nonlinear regression problems. This may invalidate direct extrapolations of linear results to nonlinear networks, such as that given by Wang and Venkatesh [5].

# 7   ACKNOWLEDGMENTS

This research was supported by NSF Presidential Young Investigator award IRI–9058450 and grant 90–21 from the James S. McDonnell Foundation to Michael C. Mozer.

## Footnotes

*Address correspondence to: dodier@cs.colorado.edu

# References

[1] Baldi, P., and Y. Chauvin. "Temporal Evolution of Generalization during Learning in Linear Networks," *Neural Computation* **3**, 589–603 (Winter 1991).

[2] Finnoff, W., F. Hergert, and H. G. Zimmermann. "Extended Regularization Methods for Nonconvergent Model Selection," in *Advances in NIPS 5*, S. Hanson, J. Cowan, and C. L. Giles, eds., pp 228–235. San Mateo, CA: Morgan Kaufmann Publishers. 1993.

[3] von Mises, R. *Mathematical Theory of Probability and Statistics*. New York: Academic Press. 1964.

[4] Morgan, N., and H. Bourlard. "Generalization and Parameter Estimation in Feedforward Nets: Some Experiments," in *Advances in NIPS 2*, D. Touretzky, ed., pp 630–637. San Mateo, CA: Morgan Kaufmann. 1990.

[5] Wang, C., and S. Venkatesh. "Temporal Dynamics of Generalization in Neural Networks," in *Advances in NIPS 7*, G. Tesauro, D. Touretzky, and T. Leen, eds. pp 263–270. Cambridge, MA: MIT Press. 1995.

[6] Wang, C., S. Venkatesh, J. S. Judd. "Optimal Stopping and Effective Machine Complexity in Learning," in *Advances in NIPS 6*, J. Cowan, G. Tesauro, and J. Alspector, eds., pp 303–310. San Francisco: Morgan Kaufmann. 1994.

[7] Weigend, A., B. Huberman, and D. Rumelhart. "Predicting the Future: A Connectionist Approach," *Int'l J. Neural Systems* **1**, 193–209 (1990).